# Multi-label Multiple Kernel Learning by Stochastic Approximation: Application to Visual Object Recognition

**Serhat S. Bucak**[*]
bucakser@cse.msu.edu

**Rong Jin**[*]
rongjin@cse.msu.edu

**Anil K. Jain**[*†]
jain@cse.msu.edu

Dept. of Comp. Sci. & Eng.[*]
Michigan State University
East Lansing, MI 48824,U.S.A.

Dept. of Brain & Cognitive Eng.[†]
Korea University, Anam-dong,
Seoul, 136-713, Korea

## Abstract

Recent studies have shown that multiple kernel learning is very effective for object recognition, leading to the popularity of kernel learning in computer vision problems. In this work, we develop an efficient algorithm for multi-label multiple kernel learning (ML-MKL). We assume that all the classes under consideration share the same combination of kernel functions, and the objective is to find the optimal kernel combination that benefits all the classes. Although several algorithms have been developed for ML-MKL, their computational cost is linear in the number of classes, making them unscalable when the number of classes is large, a challenge frequently encountered in visual object recognition. We address this computational challenge by developing a framework for ML-MKL that combines the worst-case analysis with stochastic approximation. Our analysis shows that the complexity of our algorithm is $O(m^{1/3}\sqrt{lnm})$, where $m$ is the number of classes. Empirical studies with object recognition show that while achieving similar classification accuracy, the proposed method is significantly more efficient than the state-of-the-art algorithms for ML-MKL.

## 1   Introduction

Recent studies have shown promising performance of kernel methods for object classification, recognition and localization [1]. Since the choice of kernel functions can significantly affect the performance of kernel methods, kernel learning, or more specifically Multiple Kernel Learning (MKL) [2, 3, 4, 5, 6, 7], has attracted considerable amount of interest in computer vision community. In this work, we focuss on kernel learning for object recognition because the visual content of an image can be represented in many ways, depending on the methods used for keypoint detection, descriptor/feature extraction, and keypoint quantization. Since each representation leads to a different similarity measure between images (i.e., kernel function), the related fusion problem can be cast into a MKL problem.

A number of algorithms have been developed for MKL. In [2], MKL is formulated as a quadratically constraint quadratic program (QCQP). [8] suggests an algorithm based on sequential minimization optimization (SMO) to improve the efficiency of [2]. [9] shows that MKL can be formulated as a semi-infinite linear program (SILP) and can be solved efficiently by using off-the-shelf SVM implementations. In order to improve the scalability of MKL, several first order optimization methods have been proposed, including the subgradient method [10], the level method [11], the method based on equivalence between group lasso and MKL [12, 13, 14]. Besides L1-norm [15] and L2-norm [16], Lp-norm [17] has also been proposed to regularize the weights for kernel combination. Other then the framework based on maximum margin classification, MKL can also be formulated by using kernel alignment [18] and Fisher discriminative analysis frameworks [19].

Although most efforts in MKL focus on binary classification problems, several recent studies have attempted to extend MKL to multi-class and multi-label learning [3, 20, 21, 22, 23]. Most of these studies assume that either the same or similar kernel functions are used by different but related classification tasks. Even though studies show that MKL for multi-class and multi-label learning can result in significant improvement in classification accuracy, the computational cost is often linear in the number of classes, making it computationally expensive when dealing with a large number of classes. Since most object recognition problems involve many object classes, whose number might go up to hundreds or sometimes even to thousands, it is important to develop an efficient learning algorithm for multi-class and multi-label MKL that is sublinear in the number of classes.

In this work, we develop an efficient algorithm for Multi-Label MKL (ML-MKL) that assumes all the classifiers share the same combination of kernels. We note that although this assumption significantly constrains the choice of kernel functions for different classes, our empirical studies with object recognition show that it does not affect the classification performance. A similar phenomenon was also observed in [21]. A naive implementation of ML-MKL with shared kernel combination will lead to a computational cost linear in the number of classes. We alleviate this computational challenge by exploring the idea of combining worst case analysis with stochastic approximation. Our analysis reveals that the convergence rate of the proposed algorithm is $O(m^{1/3}\sqrt{\ln m})$, which is significantly better than a linear dependence on $m$, where $m$ is the number of classes. Our empirical studies show that the proposed MKL algorithm yields similar performance as the state-of-the-art algorithms for ML-MKL, but with a significantly shorter running time, making it suitable for multi-label learning with a large number of classes.

The rest of this paper is organized as follows. Section 2 presents the proposed algorithm for Multi-Label MKL, along with its convergence analysis. Section 3 summarizes the experimental results for object recognition. Section 4 concludes this work.

## 2 Multi-label Multiple Kernel Learning (ML-MKL)

We denote by $\mathcal{D} = \{\mathbf{x}_1, \ldots, \mathbf{x}_n\}$ the collection of $n$ training instances, and by $m$ the number of classes. We introduce $\mathbf{y}^k = (y_1^k, \ldots, y_n^k)^\top \in \{-1, +1\}^n$, the assignment of the $k$th class to all the training instances: $y_i^k = +1$ if $\mathbf{x}_i$ is assigned to the $k$-th class and $y_i^k = -1$ otherwise. We introduce $\kappa_a(\mathbf{x}, \mathbf{x}') : \mathbb{R}^d \times \mathbb{R}^d \mapsto \mathbb{R}, a = 1, \ldots, s$, the $s$ kernel functions to be combined. We denote by $\{\mathbf{K}^a \in \mathbb{R}^{n \times n}, a = 1, \ldots, s\}$ the collection of $s$ kernel matrices for the data points in $\mathcal{D}$, i.e., $K_{i,j}^a = \kappa_a(\mathbf{x}_i, \mathbf{x}_j)$.

We introduce $\mathbf{p} = (p^1, \ldots, p^s)$, a probability distribution, for combining kernels. We denote by $\mathbf{K}(\mathbf{p}) = \sum_{a=1}^{s} p^a \mathbf{K}^a$ the combined kernel matrices. We introduce the domain $\mathcal{P}$ for the probability distribution $\mathbf{p}$, i.e., $\mathcal{P} = \{\mathbf{p} \in \mathbb{R}_+^s : \mathbf{p}^\top \mathbf{1} = 1\}$. Our goal is to learn from the training examples the optimal kernel combination $\mathbf{p}$ for all the $m$ classes.

The simplest approach for multi-label multiple kernel learning with shared kernel combination is to find the optimal kernel combination $\mathbf{p}$ by minimizing the sum of regularized loss functions of all $m$ classes, leading to the following optimization problem:

$$\min_{\mathbf{p} \in \mathcal{P}} \min_{\{f_k \in \mathcal{H}(\mathbf{p})\}_{k=1}^m} \left\{ \sum_{k=1}^m H_k = \sum_{k=1}^m \left\{ \frac{1}{2} |f_k|_{\mathcal{H}(\mathbf{p})}^2 + \sum_{i=1}^n \ell\left(y_i^k f_k(\mathbf{x}_i)\right) \right\} \right\}, \tag{1}$$

where $\ell(z) = \max(0, 1 - z)$ and $\mathcal{H}(\mathbf{p})$ is a Reproducing Kernel Hilbert Space endowed with kernel $\kappa(\mathbf{x}, \mathbf{x}'; \mathbf{p}) = \sum_{a=1}^s p^a \kappa_a(\mathbf{x}, \mathbf{x}')$. $H_k$ is the regularized loss function for the $k$th class. It is straightforward to verify the following dual problem of (1):

$$\min_{\mathbf{p} \in \mathcal{P}} \max_{\boldsymbol{\alpha} \in \mathcal{Q}_1} \left\{ \mathcal{L}(\mathbf{p}, \boldsymbol{\alpha}) = \sum_{k=1}^m \left\{ [\boldsymbol{\alpha}^k]^\top \mathbf{1} - \frac{1}{2}(\boldsymbol{\alpha}^k \circ \mathbf{y}^k)^\top \mathbf{K}(\mathbf{p})(\boldsymbol{\alpha}^k \circ \mathbf{y}^k) \right\} \right\}, \tag{2}$$

where $\mathcal{Q}_1 = \{\boldsymbol{\alpha} = (\boldsymbol{\alpha}^1, \ldots, \boldsymbol{\alpha}^m) : \boldsymbol{\alpha}^k \in [0, C]^n, k = 1, \ldots, m\}$. To solve the optimization problem in Eq. (2), we can view it as a minimization problem, i.e., $\min_{\mathbf{p} \in \mathcal{P}} A(\mathbf{p})$, where $A(\mathbf{p}) = \max_{\boldsymbol{\alpha} \in \mathcal{Q}_1} \mathcal{L}(\mathbf{p}, \boldsymbol{\alpha})$. We then follow the subgradient descent approach in [10] and compute the gradient of $A(\mathbf{p})$ as

$$\partial_{p^i} A(\mathbf{p}) = -\frac{1}{2} \sum_{k=1}^m (\boldsymbol{\alpha}^k(\mathbf{p}) \circ \mathbf{y}^k)^\top \mathbf{K}^i(\boldsymbol{\alpha}^k(\mathbf{p}) \circ \mathbf{y}^k),$$

where $\boldsymbol{\alpha}^k(\mathbf{p}) = \arg\max_{\boldsymbol{\alpha} \in [0,C]^n} [\boldsymbol{\alpha}^k]^\top \mathbf{1} - (\boldsymbol{\alpha}^k \circ \mathbf{y}^k)^\top \mathbf{K}(\mathbf{p})(\boldsymbol{\alpha}^k \circ \mathbf{y}^k)$. We refer to this approach as **Multi-label Multiple Kernel Learning by Sum**, or **ML-MKL-Sum**. Note that this approach is similar to the one proposed in [21]. The main computational problem with ML-MKL-Sum is that by treating every class equally, in each iteration of subgradient descent, it requires solving $m$ kernel SVMs, making it unscalable to a very large number of classes. Below we present a formulation for multi-label MKL whose computational cost is sublinear in the number of classes.

## 2.1 A Minimax Framework for Multi-label MKL

In order to alleviate the computational difficulty arising from a large number of classes, we search for the combined kernel matrix $K(\mathbf{p})$ that minimizes the worst classification error among $m$ classes, i.e.,

$$\min_{\mathbf{p} \in \mathcal{P}} \min_{\{f_k \in \mathcal{H}(\mathbf{p})\}_{k=1}^m} \max_{1 \leq k \leq m} H_k \tag{3}$$

Eq. (3) differs from Eq. (1) in that it replaces $\sum_{k=1}^m H_k$ with $\max_{1 \leq k \leq m} H_k$. The main computational advantage of using $\max_k H_k$ instead of $\sum_k H_k$ is that by using an appropriately designed method, we may be able to figure out the most difficult class in a few iterations, and spend most of the computational cycles on learning the optimal kernel combination for the most difficult class. In this way, we are able to achieve a running time that is sublinear in the number of classes. Below, we present an optimization strategy for Eq. (3) based on the idea of stochastic approximation.

A direct approach is to solve the optimization problem in Eq. (3) by its dual form. It is straightforward to derive the dual problem of Eq. (3) as follows (more details can be found in the supplementary documents)

$$\min_{\mathbf{p} \in \mathcal{P}} \max_{\boldsymbol{\beta} \in B} \left\{ \mathcal{L}(\mathbf{p}, \boldsymbol{\beta}) = \left\{ \sum_{k=1}^m \left\{ [\boldsymbol{\beta}^k]^\top \mathbf{1} - \frac{1}{2}(\boldsymbol{\beta}^k \circ \mathbf{y}^k)^\top \mathbf{K}(\mathbf{p})(\boldsymbol{\beta}^k \circ \mathbf{y}^k) \right\}^{\frac{1}{2}} \right\}^2 \right\}. \tag{4}$$

where

$$B = \left\{ (\boldsymbol{\beta}^1, \ldots, \boldsymbol{\beta}^m) : \boldsymbol{\beta}^k \in \mathbb{R}_+^n, k = 1, \ldots, m, \boldsymbol{\beta}^k \in [0, C\lambda_k]^n \text{ s.t. } \sum_{k=1}^m \lambda_k = 1 \right\}.$$

The challenge in solving Eq. (4) is that the solutions $\{\boldsymbol{\beta}^1, \ldots, \boldsymbol{\beta}^m\}$ in domain $B$ are correlated with each other, making it impossible to solve each $\boldsymbol{\beta}^k$ independently by an off-the-shelf SVM solver. Although a gradient descent approach can be developed for optimizing Eq. (4), it is unable to explore the sparse structure in $\boldsymbol{\beta}^k$ making it less efficient than state-of-the-art SVM solvers. In order to effectively explore the power of off-the-shelf SVM solvers, we rewrite (3) as follows

$$\min_{\mathbf{p} \in \mathcal{P}} \max_{\boldsymbol{\gamma} \in \Gamma} \left\{ \mathcal{L}(\mathbf{p}, \gamma) = \max_{\alpha \in \mathcal{Q}_1} \sum_{k=1}^m \gamma^k \left\{ \alpha^{k\top} \mathbf{1} - \frac{1}{2}(\boldsymbol{\alpha}^k \circ \mathbf{y}^k)^\top \mathbf{K}(\mathbf{p})(\boldsymbol{\alpha}^k \circ \mathbf{y}^k) \right\} \right\}, \tag{5}$$

where $\Gamma = \{(\gamma^1, \ldots, \gamma^m) \in \mathbb{R}_+^m : \boldsymbol{\gamma}^\top \mathbf{1} = 1\}$. In Eq. (5), we replace $\max_{1 \leq k \leq m}$ with $\max_{\boldsymbol{\gamma} \in \Gamma}$. The advantage of using Eq. (5) is that we can resort to a SVM solver to efficiently find $\boldsymbol{\alpha}^k$ for a given combination of kernels $\mathbf{K}(\mathbf{p})$.

Given Eq. (5), we develop a subgradient descent approach for solving the optimization problem. In particular, in each iteration of subgradient descent, we compute the gradient $\mathcal{L}(\mathbf{p}, \gamma)$ with respect to $\mathbf{p}$ and $\gamma$ as follows

$$\nabla_{p^a} \mathcal{L}(\mathbf{p}, \gamma) = -\frac{1}{2} \sum_{k=1}^m \gamma^k (\boldsymbol{\alpha}^k \circ \mathbf{y}^k)^\top \mathbf{K}^a (\boldsymbol{\alpha}^k \circ \mathbf{y}^k), \quad \nabla_{\gamma^k} \mathcal{L}(\mathbf{p}, \gamma) = [\boldsymbol{\alpha}^k]^\top \mathbf{1} - \frac{1}{2}(\boldsymbol{\alpha}^k \circ \mathbf{y}^k)^\top \mathbf{K}(\mathbf{p})(\boldsymbol{\alpha}^k \circ \mathbf{y}^k), \tag{6}$$

where $\boldsymbol{\alpha}^k = \arg\max_{\boldsymbol{\alpha} \in [0,C]^n} \boldsymbol{\alpha}^\top \mathbf{1} - (\boldsymbol{\alpha} \circ \mathbf{y}^k)^\top \mathbf{K}(\mathbf{p})(\boldsymbol{\alpha} \circ \mathbf{y}^k)/2$, i.e., a SVM solution to the combined kernel $\mathbf{K}(\mathbf{p})$. Following the mirror prox descent method [24], we define potential functions $\Phi_p = \frac{\eta_p}{\eta_\gamma} \sum_{a=1}^s p^a \ln p^a$ for $\mathbf{p}$ and $\Phi_\gamma = \sum_{i=1}^m \gamma^i \ln \gamma^i$ for $\boldsymbol{\gamma}$, and have the following equations for updating $\mathbf{p}_t$ and $\gamma_t$

$$p_{t+1}^a = \frac{p_t^a}{Z_t^p} \exp(-\eta_p \nabla_{p^a} \mathcal{L}(\mathbf{p}_t, \gamma_t)), \quad \gamma_{t+1}^k = \frac{\gamma_t^k}{Z_t^\gamma} \exp(-\eta_\gamma \nabla_{\gamma^k} \mathcal{L}(\mathbf{p}_t, \gamma_t)), \tag{7}$$

where $Z_t^p$ and $Z_t^\gamma$ are normalization factors that ensure $\mathbf{p}_t^\top \mathbf{1} = \boldsymbol{\gamma}_t^\top \mathbf{1} = 1$. $\eta_p > 0$ and $\eta_\gamma > 0$ are the step sizes for optimizing $\mathbf{p}$ and $\boldsymbol{\gamma}$, respectively.

Unfortunately, the algorithm described above shares the same shortcoming as the other approaches for multiple label multiple kernel learning, i.e., it requires solving $m$ SVM problems in each iteration, and therefore its computational complexity is linear in the number of classes. To alleviate this problem, we modify the above algorithm by introducing the stochastic approximation method. In particular, in each iteration $t$, instead of computing the full gradients that requirs solving $m$ SVMs, we sample one classification task according to the multinomial distribution $Multi(\gamma_t^1, \ldots, \gamma_t^m)$. Let $j_t$ be the index of the sampled classification task. Using the sampled task $j_t$, we estimate the gradient of $\mathcal{L}(\mathbf{p}, \gamma)$ with respect to $p^a$ and $\gamma^k$, denoted by $\widehat{g}_a^p(\mathbf{p}_t, \gamma_t)$ and $\widehat{g}_k^\gamma(\mathbf{p}_t, \gamma_t)$, as follows

$$\widehat{g}_a^p(\mathbf{p}_t, \gamma_t) = -\frac{1}{2}(\boldsymbol{\alpha}^{j_t} \circ \mathbf{y}^{j_t})^\top \mathbf{K}^a (\boldsymbol{\alpha}^{j_t} \circ \mathbf{y}^{j_t}), \tag{8}$$

$$\widehat{g}_k^\gamma(\mathbf{p}_t, \gamma_t) = \begin{cases} 0 & k \neq j_t \\ \frac{1}{\gamma_k}\left(\boldsymbol{\alpha}_k^\top \mathbf{1} - \frac{1}{2}(\boldsymbol{\alpha}^k \circ \mathbf{y}^k)^\top \mathbf{K}(\mathbf{p})(\boldsymbol{\alpha}^k \circ \mathbf{y}^k)\right) & k = j_t \end{cases}. \tag{9}$$

The computation of $\widehat{g}_a^p(\mathbf{p}_t, \gamma_t)$ and $\widehat{g}_i^\gamma(\mathbf{p}_t, \gamma_t)$ only requires $\boldsymbol{\alpha}^{j_t}$ and therefore only needs to solve one SVM problem, instead of $m$ SVMs. The key property of the estimated gradients in Eqs. (8) and (9) is that their expectations equal to the true gradients, as summarized by Proposition 1. This property is the key to the correctness of this algorithm.

**Proposition 1.** *We have*

$$\mathrm{E}_t[\widehat{g}_a^p(\mathbf{p}_t, \gamma_t)] = \nabla_{p_a}\mathcal{L}(\mathbf{p}_t, \gamma_t), \ \mathrm{E}_t[\widehat{g}_i^\gamma(\mathbf{p}_t, \gamma_t)] = \nabla_{\gamma_i}\mathcal{L}(\mathbf{p}_t, \gamma_t),$$

*where $E_t[\cdot]$ stands for the expectation over the randomly sampled task $j_t$.*

Given the estimated gradients, we will follow Eq. (7) for updating $\mathbf{p}$ and $\gamma$ in each iteration. Since $\widehat{g}_i^\gamma(\mathbf{p}_t, \gamma_t)$ is proportional to $1/\gamma_t$, to ensure the norm of $\widehat{g}_i^\gamma(\mathbf{p}_t, \gamma_t)$ to be bounded, we need to smooth $\gamma_{t+1}$. In order to have the smoothing effect, without modifying $\gamma_{t+1}$, we will sample directly from $\gamma'_{t+1}$,

$$\forall \gamma \in \Gamma, \exists \gamma' \in \Gamma', \text{ s.t. } \gamma'^k_{t+1} \leftarrow \gamma^k_{t+1}(1 - \delta) + \frac{\delta}{m}, k = 1, \ldots, m,$$

where $\delta > 0$ is a small probability mass used for smoothing and

$$\Gamma' = \left\{ \gamma'^\top \mathbf{1} = 1, \gamma'_k \geq \frac{\delta}{m}, k = 1, \ldots, m \right\}.$$

We refer to this algorithm as **Multi-label Multiple Kernel Learning by Stochastic Approximation**, or **ML-MKL-SA** for short. Algorithm 1 gives the detailed description.

## 2.2 Convergence Analysis

Since Eq. (5) is a convex-concave optimization problem, we introduce the following citation for measuring the quality of a solution $(\mathbf{p}, \gamma)$

$$\Delta(\mathbf{p}, \gamma) = \max_{\gamma' \in \Gamma} \mathcal{L}(\mathbf{p}, \gamma') - \min_{\mathbf{p}' \in \mathcal{P}} \mathcal{L}(\mathbf{p}', \gamma). \tag{11}$$

We denote by $(\mathbf{p}_*, \gamma_*)$ the optimal solution to Eq. (5).

**Proposition 2.** *We have the following properties for $\Delta(\mathbf{p}, \gamma)$*

1. $\Delta(\mathbf{p}, \gamma) \geq 0$ *for any solution $\mathbf{p} \in \mathcal{P}$ and $\gamma \in \Gamma$*
2. $\Delta(\mathbf{p}_*, \gamma_*) = 0$
3. $\Delta(\mathbf{p}, \gamma)$ *is jointly convex in both $\mathbf{p}$ and $\gamma$*

We have the following theorem for the convergence rate for Algorithm 1. The detailed proof can be found in the supplementary document.

**Theorem 1.** *After running Algorithm 1 over $T$ iterations, we have the following inequality for the solution $\widehat{\mathbf{p}}$ and $\widehat{\gamma}$ obtained by Algorithm 1*

$$\mathrm{E}\left[\Delta(\widehat{\mathbf{p}}, \widehat{\gamma})\right] \leq \frac{1}{\eta_\gamma T}(\ln m + \ln s) + \eta_\gamma\left(d\frac{m^2}{2\delta^2}\lambda_0^2 n^2 C^4 + n^2 C^2\right),$$

*where $d$ is a constant term, $\mathrm{E}[\cdot]$ stands for the expectation over the sampled task indices of all iterations, and $\lambda_0 = \max_{1 \leq a \leq s} \lambda_{\max}(\mathbf{K}^a)$, where $\lambda_{\max}(\mathbf{Z})$ stands for the maximum eigenvalue of matrix $\mathbf{Z}$.*

**Algorithm 1** Multi-label Multiple Kernel Learning: ML-MKL-SA

---

1: **Input**
- $\eta_p, \eta_\gamma$: step sizes
- $K^1, \ldots, K^s$: $s$ kernel matrices
- $\mathbf{y}^1, \ldots, \mathbf{y}^m$: the assignments of $m$ different classes to $n$ training instances
- $T$: number of iterations
- $\delta$: smoothing parameter

2: **Initialization**
- $\gamma_1 = \mathbf{1}/m$ and $\mathbf{p}_1 = \mathbf{1}/s$

3: **for** $t = 1, \ldots, T$ **do**

4:     Sample a classification task $j_t$ according to the distribution $Multi(\gamma_t^1, \ldots, \gamma_t^m)$.

5:     Compute $\boldsymbol{\alpha}^{j_t} = \arg\max_{\boldsymbol{\alpha} \in [0,C]^n} \boldsymbol{\alpha}^\top \mathbf{1} - (\boldsymbol{\alpha} \circ \mathbf{y}^{j_t})^\top \mathbf{K}(\mathbf{p})(\boldsymbol{\alpha} \circ \mathbf{y}^{j_t})/2$ using an off shelf SVM solver.

6:     Compute the estimated gradients $\widehat{g}_a^p(\mathbf{p}_t, \gamma_t)$ and $\widehat{g}_i^\gamma(\mathbf{p}_t, \gamma_t)$ using Eq. (8) and (9).

7:     Update $\mathbf{p}_{t+1}$, $\gamma_{t+1}$ and $\gamma_{t+1}'$ as follows

$$p_{t+1}^a = \frac{p_t^a}{Z_t^p} \exp(-\eta_\gamma \widehat{g}_a^p(\mathbf{p}_t, \gamma_t)), \ a = 1, \ldots, s.$$

$$[\gamma_{t+1}]^k = \frac{\gamma_t^k}{Z_t^\gamma} \exp(\eta_\gamma \widehat{g}_k^\gamma(\mathbf{p}_t, \gamma_t)), \ k = 1, \ldots, m; \ \gamma_{t+1}' = (1-\delta)\gamma_{t+1} + \frac{\delta}{m}\mathbf{1}.$$

8: **end for**

9: Compute the final solution $\widehat{\mathbf{p}}$ and $\widehat{\boldsymbol{\alpha}}$ as

$$\widehat{\gamma} = \frac{1}{T}\sum_{t=1}^T \gamma_t, \quad \widehat{\mathbf{p}} = \frac{1}{T}\sum_{t=1}^T \mathbf{p}_t. \tag{10}$$

---

**Corollary 1.** *With $\delta = m^{\frac{2}{3}}$ and $\eta_\gamma = \frac{1}{n}m^{-\frac{1}{3}}\sqrt{(\ln m)/T}$, after running Algorithm 1 (on the original paper) over $T$ iterations, we have $\mathrm{E}[\Delta(\widehat{\mathbf{p}}, \widehat{\gamma})] \leq O(nm^{1/3}\sqrt{(\ln m)/T})$ in terms of $m$, $n$ and $T$.*

Since we only need to solve one kernel SVM at each iteration, we have the computational complexity for the proposed algorithm on the order of $O(m^{1/3}\sqrt{(\ln m)/T})$, sublinear in the number of classes $m$.

# 3 Experiments

In this section, we empirically evaluate the proposed multiple kernel learning algorithm[2] by demonstrating its efficiency and effectiveness on the visual object recognition task.

## 3.1 Data sets

We use three benchmark data sets for visual object recognition: Caltech-101, Pascal VOC 2006 and Pascal VOC 2007. Caltech-101 contains 101 different object classes in addition to a "background" class. We use the same settings as [25] in which 30 instances of each class are used for training and 15 instances for testing. Pascal VOC 2006 data set [26] consists of $5,303$ images distributed over 10 classes, of which $2,618$ are used for training. Pascal VOC 2007 [27] consists of $5,011$ training images and $4,932$ test images that are distributed over 20 classes. For both data sets, we used the default train-test partition provided by VOC Challenge. Unlike Caltech-101 data set, where each image is assigned to one class, images in VOC data sets can be assigned to multiple classes simultaneously, making it more suitable for multi-label learning.

Table 1: Classification accuracy (AUC) and running times (second) of all ML-MKL algorithms on three data sets. Abbreviations SA, GMKL, Sum, Simple, VSKL, AVG stand for ML-MKL-SA, Generalized MKL, ML-MKL-Sum, SimpleMKL, variable sparsity kernel learning and average kernel, respectively

| dataset | Accuracy (AUC) | | | | | | Training Time (sec) | | | | | |
|---|---|---|---|---|---|---|---|---|---|---|---|---|
| | SA | GMKL | Sum | Simple | VSKL | AVG | SA | GMKL | Sum | Simple | VSKL | AVG |
| CALTECH-101 | 0.80 | 0.79 | 0.80 | 0.78 | 0.77 | 0.77 | 191.17 | 18292.00 | 1814.50 | 9869.40 | 21266.05 | N/A |
| VOC2006 | 0.75 | 0.75 | 0.74 | 0.74 | 0.74 | 0.72 | 245.10 | 2586.90 | 890.65 | 11549.00 | 7368.27 | N/A |
| VOC2007 | 0.50 | 0.49 | 0.47 | 0.42 | 0.46 | 0.45 | 1329.40 | 30333.14 | 1372.60 | 18536.37 | 11370.48 | N/A |

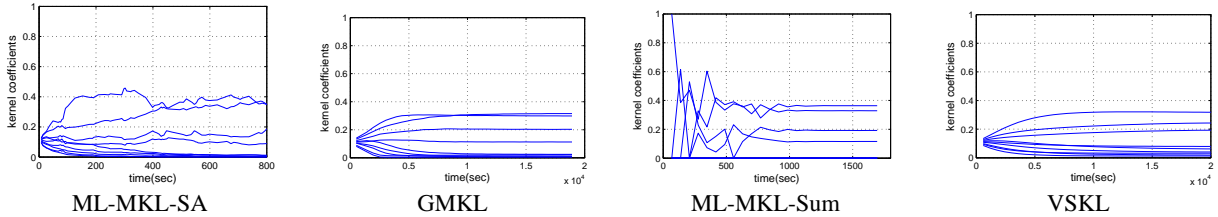

ML-MKL-SA  GMKL  ML-MKL-Sum  VSKL

Figure 1: The evolution of kernel weights over time for CALTECH-101 data set. For GMKL and VSKL, the curves display the kernel weights that are averaged over all the classes since a different kernel combination is learnt for each class.

## 3.2 Kernels

We extracted 9 kernels for Caltech-101 data set by using the software provided in [28]. Three different feature extraction methods are used for kernel construction: (i) GB: geometric blur descriptors are applied to the detected keypoints [29]; RBF kernel is used in which the distance between two images is computed by averaging the distance of the nearest descriptor pairs for the image pair. (ii) PHOW gray/color: keypoints based on dense sampling; SIFT descriptors are quantized to 300 words and spatial histograms with 2x2 and 4x4 subdivisions are built to generate chi-squared kernels [30]. (iii) SSIM: self-similarity features taken from [31] are used and spatial histograms based on 300 visual words are used to form the chi-squared kernel.

For VOC data sets, a different procedure, based on the reports of VOC challenges [1], is used to construct multiple visual dictionaries, and each dictionary results in a different kernel. To obtain multiple visual dictionaries, we deploy (i) three keypoint detectors, i.e., dense sampling, HARHES [32] and HESLAP [33], (ii) two keypoint descriptors, i.e., SIFT [33] and SPIN [34]), (iii) two different numbers of visual words, i.e., 500 and 1,000 visual words, (iv) two different kernel functions, i.e., linear kernel and chi-squared kernel. The bandwidth of the chi-squared kernels is calculated using the procedure in [25]. Using the above variants in visual dictionary construction, we constructed 22 kernels for both VOC2007 and VOC2006 data sets. In addition to the K-means implementation in [28], we also applied a hierarchical clustering algorithm [35] to descriptor quantization for VOC 2007 data set, leading to four more kernels for VOC2007 data set.

## 3.3 Baseline Methods

We first compare the proposed algorithm ML-MKL-SA to the following MKL algorithms that learn a different kernel combination for each class: (i) Generalized multiple kernel learning method (GMKL) [25], which reports promising results for object recognition, (ii) SimpleMKL [10], which learns the kernel combination by a subgradient approach and (iii) Variable Sparsity Kernel Learning (VSKL), a miror-prox descent based algorithm for MKL [36]. We also compare ML-MKL-SA to ML-MKL-Sum, which learns a kernel combination shared by all classes as described in Section 2 using the optimization method in [21]. In all implementations of ML multiple kernel learning algorithms,we use LIBSVM implementation of one-versus-all SVM where needed.

## 3.4 Experimental Results

To evaluate the effectiveness of different algorithms for multi-label multiple kernel learning, we first compute the area under precision-recall curve (AUC) for each class, and report the value of AUC averaged over all the classes. We

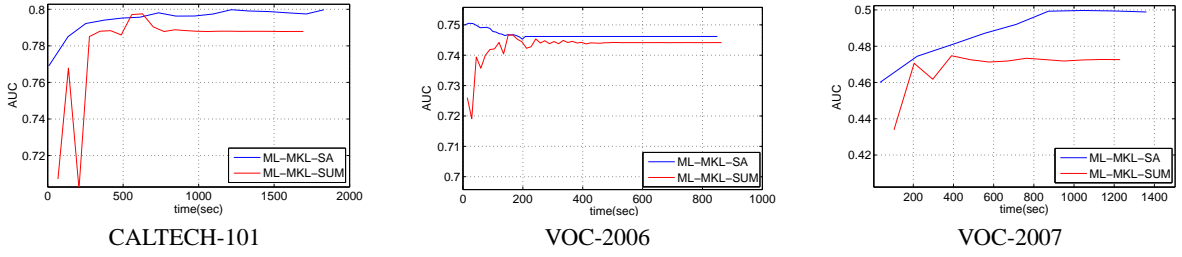

CALTECH-101          VOC-2006          VOC-2007

Figure 2: The evolution of classification accuracy over time for ML-MKL-SA and ML-MKL-Sum on three data sets

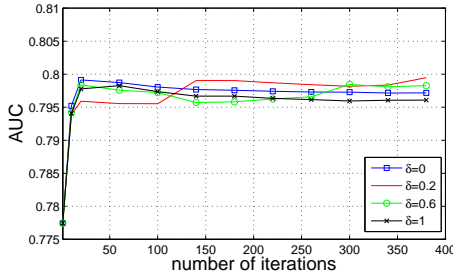

Figure 3: Classification accuracy (AUC) of the proposed algorithm Ml-MKL-SA on CALTECH-101 using different values of $\delta$ (for $\eta_p = \eta_\gamma = 0.01$).

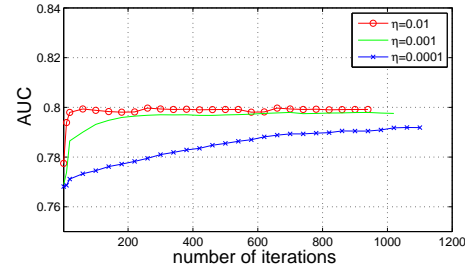

Figure 4: Classification accuracy (AUC) of the proposed algorithm Ml-MKL-SA on CALTECH-101 using different values of $\eta_p = \eta_\gamma = \eta$ for ($\delta = 0$).

evaluate the efficiency of algorithms by their running times for training. All methods are coded in MATLAB and are implemented on machines with 2 dual-core AMD Opterons running at 2.2GHz, 8GB RAM and linux operating system.

For the proposed method, itarations stop when $\frac{\widehat{\mathbf{p}}_t - \widehat{\mathbf{p}}_{t-1}}{\widehat{\mathbf{p}}_t}$ is smaller than 0.01. Unless stated, the smoothing parameter $\delta$ is set to be 0.2. For simplicity we take $\eta = \eta_p = \eta_\gamma$ in all the following experiments. Step size $\eta$ is chosen as 0.0001 for CALTECH-101 data set and 0.001 for VOC data sets in order to achieve the best computational efficiency.

Table 1 summarizes the classification accuracies (AUC) and the running times of all the algorithms over the three data sets. We first note that the proposed MKL method for multi-labeled data, i.e., ML-MKL-SA, yields the best performance among the methods in comparison, which justifies the assumption of using the same kernel combination for all the classes. Note that a simple approach that uses the average of all kernels yields reasonable performance, although its classification accuracy is significantly worse than the proposed approach ML-MKL-SA. Second, we observe that except for the average kernel method that does not require learning the kernel combination weights, ML-MKL-SA and ML-MKL-Sum are significantly more efficient than the other baseline approaches. This is not surprising as ML-MKL-SA and ML-MKL-Sum compute a single kernel combination for all classes. Third, compared to ML-MKL-Sum, we observe that ML-MKL-SA is overall more efficient, and significantly more efficient for CALTECH-101 data set. This is because the number of classes in CALTECH-101 is significantly larger than that of the two VOC challenge data sets. This result further confirms that the proposed algorithm is scalable to the data sets with a large number of classes.

Fig. 1 shows the change in the kernel weights over time for the proposed method and the three baseline methods (i.e., ML-MKL-Sum, GMKL, and VSKL) on CALTECH-101 data set. We observe that, overall, ML-MKL-SA shares a similar pattern as GMKL and VSKL in the evolution curves of kernel weights, but is ten times faster than the two baseline methods. Although ML-MKL-Sum is significantly more efficient than GMKL and VSKL, the kernel weights learned by ML-MKL-Sum vary significantly, particularly at the beginning of the learning process, making it a less stable algorithm than the proposed algorithm ML-MKL-SA. To further compare ML-MKL-SA with ML-MKL-Sum, in Fig. 2, we show how the classification accuracy is changed over time for both methods for all three data sets. We again observe the unstable behavior of ML-MKL-Sum: the classification accuracy of ML-MKL-Sum could vary significantly over a relatively short period of time, making it less desirable method for MKL.

To evaluate the sensitivity of the proposed method to parameters $\delta$ and $\eta$, we conducted experiments with varied values for the two parameters. Fig. 3 shows how the classification accuracy (AUC) of the proposed algorithm changes over iterations on CALTECH-101 using four different values of $\delta$. We observe that the final classification accuracy is comparable for different values of $\delta$, demonstrating the robustness of the proposed method to the choice of $\delta$. We also note that the two extreme cases, i.e, $\delta = 0$ and $\delta = 1$, give the worst performance, indicating the importance of choosing an optimal value for $\delta$. Fig. 4 shows the classification accuracy for three different values of $\eta$ on CALTECH-101 data set. We observe that the proposed algorithm achieves similar classification accuracy when $\eta$ is set to be a relatively small value (i.e., $\eta = 0.001$ and $\eta = 0.0001$). This result demonstrates that the proposed algorithm is in general insensitive to the choice of step size ($\eta$).

## 4 Conclusion and Future Work

In this paper, we present an efficient optimization framework for multi-label multiple kernel learning that combines worst-case analysis with stochastic approximation. Compared to the other algorithms for ML-MKL, the key advantage of the proposed algorithm is that its computational cost is sublinear in the number of classes, making it suitable for handling a large number of classes. We verify the effectiveness of the proposed algorithm by experiments in object recognition on several benchmark data sets. There are two directions that we plan to explore in the future. First, we aim to further improve the efficiency of ML-MKL by reducing its dependence on the number of training examples and speeding up the convergence rate. Second, we plan to improve the effectiveness and efficiency of multi-label learning by exploring the correlation and structure among the classes.

## 5 Acknowledgements

This work was supported in part by National Science Foundation (IIS-0643494), US Army Research (ARO Award W911NF-08-010403) and Office of Naval Research (ONR N00014-09-1-0663). Any opinions, findings and conclusions or recommendations expressed in this material are those of the authors and do not necessarily reflect the views of NFS, ARO, and ONR. Part of Anil Jain's research was supported by WCU (World Class University) program through the National Research Foundation of Korea funded by the Ministry of Education, Science and Technology (R31-2008-000-10008-0).

## Footnotes

[2]Codes can be downloaded from http://www.cse.msu.edu/~bucakser/ML-MKL-SA.rar

## References

[1] M. Everingham, L. Van Gool, C. K. I. Williams, J. Winn, and A. Zisserman, "The PASCAL Visual Object Classes Challenge 2009 (VOC2009) Results." http://www.pascal-network.org/challenges/VOC/voc2009/workshop/index.html.

[2] G. Lanckriet, T. De Bie, N. Cristianini, M. Jordan, and W. Noble, "A statistical framework for genomic data fusion," *Bioinformatics*, vol. 20, pp. 2626–2635, 2004.

[3] S. Ji, L. Sun, R. Jin, and J. Ye, "Multi-label multiple kernel learning," in *Proceedings of Neural Information Processings Systems*, 2008.

[4] G. Lanckriet, N. Cristianini, P. Bartlett, L. Ghaoui, and M. Jordan, "Learning the kernel matrix with semidefinite programming," *Journal of Machine Learning Research*, vol. 5, pp. 27–72, 2004.

[5] O. Chapelle and A. Rakotomamonjy, "Second order optimization of kernel parameters," in *NIPS Workshop on Kernel Learning: Automatic Selection of Optimal Kernels*, 2008.

[6] P. Gehler and S. Nowozin, "On feature combination for multiclass object classification," in *Proceedings of the IEEE International Conference on Computer Vision*, 2009.

[7] P. Gehler and S. Nowozin, "Let the kernel figure it out: Principled learning of pre-processing for kernel classifiers," in *Proceedings of the IEEE Conference on Computer Vision and Pattern Recognition*, 2009.

[8] F. Bach, G. Lanckriet, and M. Jordan, "Multiple kernel learning, conic duality, and the smo algorithm," in *Proceedings of the 21st International Conference on Machine Learning*, 2004.

[9] S. Sonnenburg, G. Ratsch, and C. Schafer, "A general and efficient multiple kernel learning algorithm," in *Proceedings of Neural Information Processings Systems*, pp. 1273–1280, 2006.

[10] A. Rakotomamonjy, F. Bach, Y. Grandvalet, and S. Canu, "SimpleMKL," *Journal of Machine Learning Research*, vol. 9, pp. 2491–2521, 2008.

[11] Z. Xu, R. Jin, I. King, and M. R. Lyu, "An extended level method for efficient multiple kernel learning," in *Proceedings of Neural Information Processings Systems*, pp. 1825–1832, 2008.

[12] Z. Xu, R. Jin, H. Yang, I. King, and M. R. Lyu, "Simple and efficient multiple kernel learning by group lasso," in *Proceedings of the 27th International Conference on Machine Learning*, 2010.

[13] F. Bach, "Consistency of the group lasso and multiple kernel learning," *Journal of Machine Learning Research*, vol. 9, pp. 1179–1225, 2008.

[14] Z. Xu, R. Jin, S. Zhu, M. R. Lyu, and I. King, "Smooth optimization for effective multiple kernel learning," in *Proceedings of the AAAI Conference on Artificial Intelligence*, 2010.

[15] A. Rakotomamonjy, F. Bach, S. Canu, and Y. Grandvalet, "More efficiency in multiple kernel learning," in *Proceedings of the 24th International Conference on Machine Learning*, 2007.

[16] M. Kloft, U. Brefeld, A. Sonnenburg, and A. Zien, "Comparing sparse and non-sparse multiple kernel learning," in *NIPS Workshop on Understanding Multiple Kernel Learning Methods*, 2009.

[17] M. Kloft, U. Brefeld, A. Sonnenburg, P. Laskov, K.-R. Muller, and A. Zien, "Efficient and accurate lp-norm multiple kernel learning," in *Proceedings of Neural Information Processings Systems*, 2009.

[18] S. Hoi, M. Lyu, and E. Chang, "Learning the unified kernel machines for classification," in *Proceedings of the Conference on Knowledge Discovery and Data Mining*, p. 187196, 2006.

[19] J. Ye, J. Chen, and J. S., "Discriminant kernel and regularization parameter learning via semidefinite programming," in *Proceedings of the International Conference on Machine Learning*, p. 10951102, 2007.

[20] A. Zien and S. Cheng, "Multiclass multiple kernel learning," in *Proceedings of the 24th International Conference on Machine Learning*, 2007.

[21] L. Tang, J. Chen, and J. Ye, "On multiple kernel learning with multiple labels," in *Proceedings of the 21st International Jont Conference on Artifical Intelligence*, 2009.

[22] J. Yang, Y. Li, Y. Tian, L. Duan, and W. Gao, "Group-sensitive multiple kernel learning for object categorization," in *Proceedings of the IEEE International Conference on Computer Vision*, 2009.

[23] F. Orabona, L. Jie, and B. Caputo, "Online-batch strongly convex multi kernel learning," in *Proceedings of the IEEE Conference on Computer Vision and Pattern Recognition*, 2010.

[24] A. Nemirovski, "Prox-method with rate of convergence o(1/t) for variational inequalities with lipschitz continuous monotone operators and smooth convex-concave saddle point problems," *SIAM Journal on Optimization*, vol. 15, pp. 229–251, 2004.

[25] M. Varma and D. Ray, "Learning the discriminative power-invariance trade-off," in *Proceedings of the IEEE International Conference on Computer Vision*, October 2007.

[26] M. Everingham, A. Zisserman, C. K. I. Williams, and L. Van Gool, "The PASCAL Visual Object Classes Challenge 2006 (VOC2006) Results." http://www.pascal-network.org/challenges/VOC/voc2006/results.pdf.

[27] M. Everingham, L. Van Gool, C. K. I. Williams, J. Winn, and A. Zisserman, "The PASCAL Visual Object Classes Challenge 2007 (VOC2007) Results." http://www.pascal-network.org/challenges/VOC/voc2007/workshop/index.html.

[28] A. Vedaldi and B. Fulkerson, "VLFeat: An open and portable library of computer vision algorithms." http://www.vlfeat.org/, 2008.

[29] A. Berg, T. Berg, and J. Malik, "Shape matching and object recognition using low distortion correspondences," in *Proceedings of the IEEE Conference on Computer Vision and Pattern Recognition*, 2005.

[30] S. Lazebnik, C. Schmid, and P. Ponce, "Beyond bag of features: Spatial pyramid matching for recognizing natural scene categories," in *Proceedings of the IEEE Conference on Computer Vision and Pattern Recognition*, 2006.

[31] E. Shechtman and I. M., "Matching local self-similarities across images and videos," in *Proceedings of the IEEE Conference on Computer Vision and Pattern Recognition*, 2007.

[32] K. Mikolajczyk and C. Schmid, "Distinctive image features from scale-invariant keypoints," *IEEE Transactions on Pattern Analysis and Machine Intelligence*, vol. 27, no. 10, pp. 1615–1630, 2005.

[33] D. Lowe, "Distinctive image features from scale-invariant keypoints," *International Journal of Computer Vision*, vol. 2, no. 60, pp. 91–110, 2004.

[34] S. Lazebnik, C. Schmid, and P. Ponce, "Sparse texture representation using affine-invariant neighborhoods," in *Proceedings of the IEEE Conference on Computer Vision and Pattern Recognition*, 2003.

[35] M. Muja and D. G. Lowe, "Fast approximate nearest neighbors with automatic algorithm configuration," in *Proceedings of the International Conference on Computer Vision Theory and Application*, pp. 331–340, INSTICC Press, 2009.

[36] J. Saketha Nath, G. Dinesh, S. Raman, C. Bhattacharyya, A. Ben-Tal, and K. Ramakrishan, "On the algorithmics and applications of a mixed-norm based kernel learning formulation," in *Proceedings of Neural Information Processings Systems*, 2009.

